# A Computational Model of Prefrontal Cortex Function

**Todd S. Braver**
Dept. of Psychology
Carnegie Mellon Univ.
Pittsburgh, PA 15213

**Jonathan D. Cohen**
Dept. of Psychology
Carnegie Mellon Univ.
Pittsburgh, PA 15213

**David Servan-Schreiber**
Dept. of Psychiatry
Univ. of Pittsburgh
Pittsburgh, PA 15232

## Abstract

Accumulating data from neurophysiology and neuropsychology have suggested two information processing roles for prefrontal cortex (PFC): 1) short-term active memory; and 2) inhibition. We present a new behavioral task and a computational model which were developed in parallel. The task was developed to probe both of these prefrontal functions simultaneously, and produces a rich set of behavioral data that act as constraints on the model. The model is implemented in continuous-time, thus providing a natural framework in which to study the temporal dynamics of processing in the task. We show how the model can be used to examine the behavioral consequences of neuromodulation in PFC. Specifically, we use the model to make novel and testable predictions regarding the behavioral performance of schizophrenics, who are hypothesized to suffer from reduced dopaminergic tone in this brain area.

## 1 Introduction

Prefrontal cortex (PFC) is an area of the human brain which is significantly expanded relative to other animals. There is general consensus that the PFC is centrally involved in higher cognitive activities such as planning, problem solving and language. Recently, the PFC has been associated with two specific information processing mechanisms: short-term active memory and inhibition. Active memory is the capacity of the nervous system to maintain information in the form of sustained activation states (e.g., cell firing) for short periods of time. This can be distinguished from forms of memory that are longer in duration and are instantiated as

modified values of physiological parameters (e.g., synaptic strength). Over the last two decades, there have been a large number of neurophysiological studies focusing on the cellular basis of active memory in prefrontal cortex. These studies have revealed neurons in PFC that fire selectively to specific stimuli and response patterns, and that remain active during a delay between these. Investigators such as Fuster (1989) and Goldman-Rakic (1987) have argued from this data that PFC maintains temporary information needed to guide behavioral responses through sustained patterns of neural activity. This hypothesis is consistent with behavioral findings from both animal and human lesion studies, which suggest that PFC is required for tasks involving delayed responses to prior stimuli (Fuster, 1989; Stuss & Benson, 1986).

In addition to its role in active memory, many investigators have focused on the inhibitory functions of PFC. It has been argued that PFC representations are required to overcome reflexive or previously reinforced response tendencies in order to mediate a contextually appropriate – but otherwise weaker – response (Cohen & Servan-Schreiber, 1992). Clinically, it has been observed that lesions to PFC are often associated with a syndrome of behavioral disinhibition, in which patients act in impulsive and often socially inappropriate ways (Stuss & Benson, 1986). This syndrome has often been cited as evidence that PFC plays an important role inhibiting behaviors which are compelling but socially inappropriate.

While the involvement of PFC in both active memory and inhibition is generally agreed upon, computational models can play an important role in providing mechanisms by which to explain how these two information processing functions arise. There are several computational models now in the literature which have focused on either the active memory (Zipser, 1991), or inhibitory (Levine & Pruiett, 1989) functions of PFC, or both functions together (Dehaene & Changeux, 1989; Cohen & Servan-Schreiber, 1992). These models have been instrumental in explaining the role of PFC in a variety of behavioral tasks (e.g., the Wisconsin Card Sort and Stroop). However, these earlier models are limited by their inability to fully capture the dynamical processes underlying active memory and inhibition. Specifically, none of the simulations have been tightly constrained by the temporal parameters found in the behavioral tasks (e.g., durations of stimuli, delay periods, and response latencies). This limitation is not found solely in the models, but is also a feature of the behavioral tasks themselves. The tasks simulated were not structured in ways that could facilitate a dynamical analysis of processing.

In this paper we address the limitations of the previous work by describing both a new behavioral task and a computational model of PFC. These have been developed in parallel and, together, provide a useful framework for exploring the temporal dynamics of active memory and inhibition and their consequences for behavior. We then go on to describe how this framework can be used to examine neuromodulatory effects in PFC, which are believed to play a critical role in both normal functioning and in psychiatric disorders, such as schizophrenia.

## 2    Behavioral Assessment of Human PFC Function

We have developed a task paradigm which incorporates two components central to the function of prefrontal cortex – short-term active memory and inhibition – and that can be used to study the dynamics of processing. The task is a variant of the continuous performance test (CPT), which is commonly used to study attention in

behavioral and clinical research. In a standard version of the task (the CPT-AX), letters are presented one at a time in the middle of a computer screen. Subjects are instructed to press the target button to the letter X (probe stimulus) but only when it is preceded by an A (the cue stimulus). In previous versions of the CPT, subjects only responded on target trials. In the present version of the task, a two response forced-choice procedure is employed; on non-A-X trials subjects are asked to press the non-target button. This procedure allows for response latencies to be evaluated on every trial, thus providing more information about the temporal dimensions of processing in the task.

Two additional modifications were made to the standard paradigm in order to maximally engage PFC activity. The memory function of PFC is tapped by manipulating the delay between stimuli. In the CPT-AX, the prior stimulus (cue or non-cue) provides the context necessary to decide how to respond to the probe letter. However, with a short delay (750 msec.), there is little demand on memory for the prior stimulus. This is supported by evidence that PFC lesions have been shown to have no effect on performance when there is only a short delay (Stuss & Benson, 1986). With a longer delay (5000 msec.), however, it becomes necessary to maintain a representation of the prior stimulus in order for it to be used as context for responding to the current one. The ability of the PFC to sustain contextual representations over the delay period can be determined behaviorally by comparing performance on short delay trials (50%) against those with long delays (50%).

The inhibitory function of PFC is probed by introducing a prepotent response tendency that must be overcome to respond correctly. This tendency is introduced into the task by increasing the frequency of target trials (A followed by X). In the remaining trials, there are three types of distractors: 1) a cue followed by a non-target probe letter (e.g., A-Y); 2) a non-cue followed by the target probe letter (e.g., B-X); and a non-cue followed by a non-target probe letter (e.g., B-Y). Target trials occur 70% of the time, while each type of distractor trial occurs only 10% of the time. The frequency of targets promotes the development of a strong tendency to respond to the target probe letter whenever it occurs, regardless of the identity of the cue (since a response to the X itself is correct 7 out of 8 times).

The ability to inhibit this response tendency can be examined by comparing accuracy on trials when the target occurs in the absence of the cue (B-X trials), with those made when neither the cue nor target occurs (i.e., B-Y trials, which provide a measure of non-specific response bias and random responding). Trials in which the cue but not the target probe appears (A-Y trials) are also particularly interesting with respect to PFC function. These trials measure the cumulative influence of active representations of context in guiding responses. In a normally functioning system, context representations should stabilize and increase in strength as time progresses. Thus, it is expected that A-Y accuracy will tend to decrease for long delay trials relative to short ones.

As mentioned above, the primary benefit of this paradigm is that it provides a framework in which to simultaneously probe the inhibitory and memory functions associated with PFC. This is supported by preliminary neuroimaging data from our laboratory (using PET) which suggests that PFC is, in fact, activated during performance of the task. Although it is simple in structure, the task also generates a rich set of behavioral data. There are four stimulus conditions crossed with two delay conditions for which both accuracy and reaction time performance can be

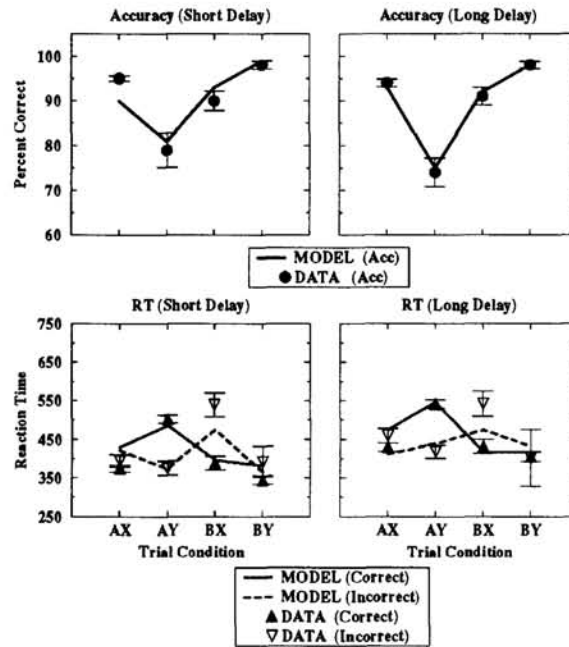

**Figure 1:** Subject behavioral data with model performance superimposed. **Top Panels:** Accuracy across both delays in all four conditions. **Bottom Panels:** Reaction times for both correct and incorrect responses in all conditions. Bars represent standard error of measurement for the empirical data.

measured. Figure 1 shows data gathered from 36 college-age subjects performing this task.

In brief, we found that: 1) Accuracy was relatively unchanged in the long delays compared to the short, demonstrating that active memory was adequately supporting performance; 2) A-Y accuracy, however, did slightly decrease at long delays, reflecting the normal build-up of context representations over time; 3) Accuracy on B-X trials was relatively high, supporting the assumption that subjects could effectively use context representations to inhibit prepotent responses; 4) A distinct pattern emerged in the latencies of correct and incorrect responses, providing information on the temporal dynamics of processing (i.e., responses to A-Y trials are slow on correct trials and fast on incorrect ones; the pattern is reversed for B-X trials). Taken together, the data provides specific, detailed information about normal PFC functioning, which act as constraints on the development and evaluation of a computational model.

## 3   A Computational Model of the CPT-AX

We have developed a recurrent network model which produces detailed information regarding the temporal course of processing in the CPT-AX task. The network is composed of three modules: an input module, a memory module, and an output module. The memory module implements the memory and inhibitory functions believed to be carried out by PFC. Figure 2 shows a diagram of the model.

Each unit in the input module represents a different stimulus condition: A, B, X &

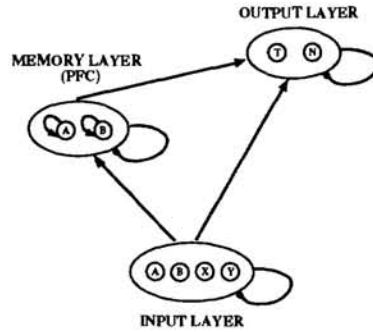

Figure 2: A diagram of the CPT-AX model.

Y. Units in the input module make excitatory connections on the response module, both directly and indirectly through the memory module. Lateral inhibition within each layer produces competition for representations. Activity from the cue stimulus flows to the memory module, which is responsible for maintaining a trace of the relevant context in each trial. Units in the memory module have self-excitatory connections, which allow for the activity generated by the cue to be sustained in the absence of input. The recurrent connectivity utilized by each unit in this module is assumed to be a simpler, but formally equivalent analogue of a fully connected recurrent cell assembly. Further, Zipser (1991) has used this type of connectivity to produce temporal activity patterns which are highly similar to the firing patterns of neurons in memory-associated areas of cortex, such as PFC. Activity from the input and memory modules is integrated in the output module. The output of this module determines whether a target (T) or non-target (N) response is made.

To simulate the CPT-AX task we have purposefully kept the network architecture and size as simple as possible in order to maximize the model's interpretability. We have therefore not attempted to simulate neural information processing in a neuron-by-neuron manner. Rather, the populations of a few units are seen as capturing the information processing characteristics of much larger populations of real neurons. In this way, it is possible to capture the stochastic, distributed, and dynamical properties of real neural networks with small and analytically tractable simulations.

The simulation is run in a temporally continuous framework in which processing is governed by the following difference equation:

$$I_j(t+1) = (\gamma \sum_i w_{ij} y_i + \beta - I_j(t))dt \tag{1}$$

where

$$y_j = \frac{1}{1 + e^{-I_j}} \tag{2}$$

is the state of unit j, $I_j$ is the total input to j, $dt$ is the time-step of integration, $\gamma$ is the gain and $\beta$ is the bias. The continuous framework is preferable to a discrete event-based one in that it allows for a plausible way to scale events appropriately to the exact temporal specifications of the task (i.e., the duration of stimuli and the delay between cue and probe). In addition, the continuous character of the simulation naturally provides a framework for inferring the reaction times in the various conditions.

## 4  Simulations of Behavioral Performance

We used a continuous recurrent generalization of backpropagation (Pearlmutter, 1989) to train the network to perform the CPT-AX. All of the connection weights were developed entirely by the training procedure, with the constraint that that all self and between layer weights were forced to be positive and all within layer weights were forced to be negative. Training consisted of repeated presentation of each of the 8 conditions in the task (A-X,A-Y,B-X,B-Y, at both long and short delays), with the presentation frequency of each condition matching that of the behavioral task. Weights were updated after the presentation of each trial, biases ($\beta$) were fixed at -2.5, and $dt$ was set at 0.1. The network was trained deterministically; completion of training occurred when network accuracy reached 100% for each condition.

Following training, weights were fixed. Errors and reaction time distributions were then simulated by adding zero-mean Gaussian noise to the net input of each unit at every time step during trial presentation. A trial consisted of the presentation of the cue stimulus, a delay period and then the probe stimulus. As mentioned above, the duration of these events was appropriately scaled to match the temporal parameters of the task (e.g., 300 msec. duration for cue and probe presentation, 750 msec. for short delays, 5000 msec. for long delays). A time constant ($\tau$) of 50 msec. was used for simulation in the network. This scaling factor provided sufficient temporal resolution to capture the relationship between the two task delays while still permitting a tractable way of simulating the events.

Responses were determined by noting which output unit reached a threshold value first following presentation of the probe stimulus. Response latency was determined by calculating the number of time steps taken by the model to reach threshold multiplied by the time constant $\tau$. To facilitate comparisons with the experimental reaction times, a constant k was added to all values produced. This parameter might correspond to the time required to execute a motor response. The value of k was determined by a least mean squares fit to the data. 1000 trials of each condition were run in order to obtain a reliable estimate of performance under stochastic conditions. The standard deviation of the noise distribution ($\sigma$) and the threshold (T) of the response units were adjusted to produce the best fit to the subject data. Figure 1 compares the results of the simulation against the behavioral data.

As can be seen in the figure, the model provides a good fit to the behavioral data in both the pattern of accuracy and reaction times. The model not only matches the qualitative pattern of errors and reaction times but produces very similar quantitative results as well. The match between model and experimental results is particularly striking when it is considered that there are a total of 24 data points that this model is fitting, with only 4 free parameters ($\sigma$,T,$\tau$,k). The model's ability to successfully account for the pattern of behavioral performance provides convincing evidence that it captures the essential principles of processing in the task. We can then feel confident in not only examining normal processing, but also in extending the model to explore the effects of specific disturbances to processing in PFC.

## 5  Behavioral Effects of Neuromodulation in PFC

In a previous meeting of this conference a simulation of a simpler version of the CPT was discussed (Servan-Schreiber, Printz, & Cohen, 1990). In this simulation the

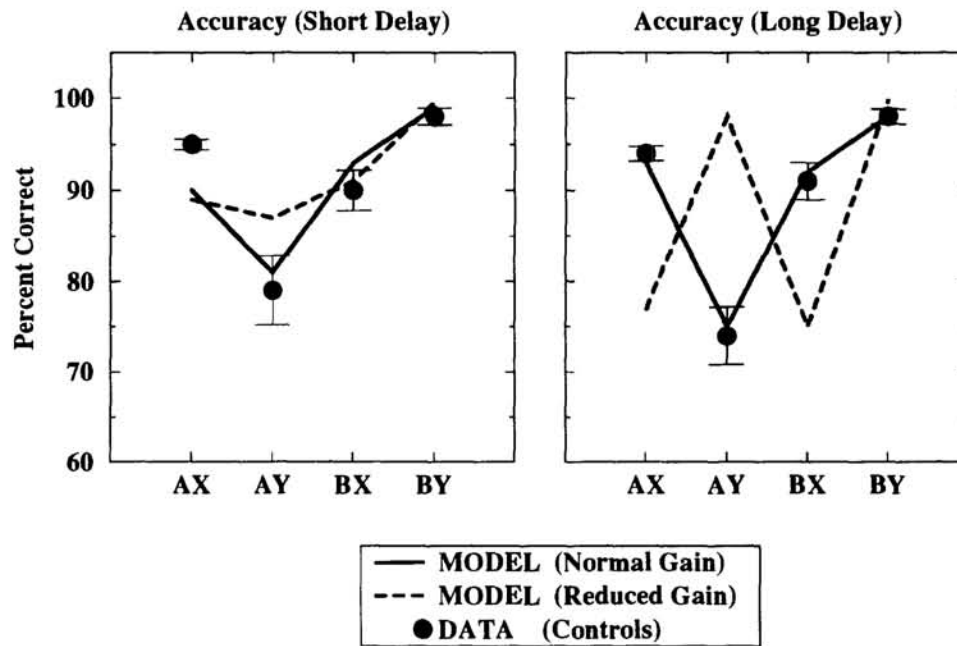

Figure 3: Comparision of of model performance with normal and reduced gain. The graph illustrates the effect of reducing gain in the memory layer on task performance. In the baseline network $\gamma=1$, in the reduced-gain network $\gamma=0.8$.

effects of system-wide changes in catecholaminergic tone were captured by changing the gain ($\gamma$) parameter of network units. Changes in gain are thought correspond to the action of modulatory neurotransmitters in modifying the responsivity of neurons to input signals (Servan-Schreiber et al., 1990; Cohen & Servan-Schreiber, 1992).

The current simulation of the CPT offers the opportunity to explore the effects of neuromodulation on the information processing functions specific to PFC. The transmitter dopamine is known to modulate activity in PFC, and manipulations to prefrontal dopamine have been shown to have effects on both memory-related neuronal activity and behavioral performance (Sawaguchi & Goldman-Rakic, 1991). Furthermore, it has been hypothesized that reductions of the neuromodulatory effects of dopamine in PFC are responsible for some of the information processing deficits seen in schizophrenia. To simulate the behavior of schizophrenic subjects, we therefore reduce the gain ($\gamma$) of units in the memory module of the network.

With reduced gain in the memory module, there are striking changes in the model's performance of the task. As can be seen in Figure 3, in the short delay conditions the performance of the reduced-gain model is relatively similar to that of control subjects (and the intact model). However, at long delays, the reduced-gain model produces a qualitatively different pattern of performance. In this condition, the model has a high B-X error rate but a low A-Y error rate, a pattern which is opposite to that seen in the control subjects. This double dissociation in performance is a robust effect of the reduced-gain simulation (i.e., it seems relatively uninfluenced by other parameter adjustments).

Thus, the model makes clear-cut predictions which are both novel and highly testable. Specifically, the model predicts that: 1) Differences in performance be-

tween control and schizophrenic subjects will be most apparent at long delays; 2) Schizophrenics will perform significantly worse than control subjects on B-X trials at long delays; 3) Schizophrenics will perform significantly *better* than control subjects on A-Y trials at long delays. This last prediction is especially interesting given the fact that tasks in which schizophrenics show superior performance relative to controls are relatively rare in experimental research.

Furthermore, the model not only makes predictions regarding schizophrenic behavioral performance, but also offers explanations as to their mechanisms. Analyses of the trajectories of activation states in the memory module reveals that both of the dissociations in performance are due to failures in maintaining representations of the context set up by the cue stimulus. Reducing gain in the memory module blurs the distinction between signal and noise, and causes the context representations to decay over time. As a result, in the long delay trials, there is a higher probability that the model will show both failures of inhibition (more B-X errors) and memory (less A-Y errors).

## 6 Conclusions

The results of this paper show how a computational analysis of the temporal dynamics of PFC information processing can aid in understanding both normal and disturbed behavior. We have developed a behavioral task which simultaneously probes both the inhibitory and active memory functions of PFC. We have used this task in combination with a computational model to explore the effects of neuromodulatory dysfunction, making specific predictions regarding schizophrenic performance in the CPT-AX. Confirmation of these predictions now await further testing.

## References

Cohen, J. & Servan-Schreiber, D. (1992). Context, cortex, and dopamine: A connectionist approach to behavior and biology in schizophrenia. *Psychological Review, 99*, 45–77.

Dehaene, S. & Changeux, J. (1989). A simple model of prefrontal cortex function in delayed-response tasks. *Journal of Cognitive Neuroscience, 1*(3), 244–261.

Fuster, J. (1989). *The prefrontal cortex*. New York: Raven Press.

Goldman-Rakic, P. (1987). Circuitry of primate prefrontal cortex and regulation of behavior by representational memory. In F. Plum (Ed.), *Handbook of physiology-the nervous system, v.* Bethesda, MD: American Physiological Society, 373–417.

Levine, D. & Pruiett, P. (1989). Modeling some effects of frontal lobe damage: novelty and perseveration. *Neural Networks, 2*, 103–116.

Pearlmutter, B. (1989). Learning state space trajectories in recurrent neural networks. *Neural Computation, 1*, 263–269.

Sawaguchi, T. & Goldman-Rakic, P. (1991). D1 dopamine receptors in prefrontal cortex: Involvement in working memory. *Science, 251*, 947–950.

Servan-Schreiber, D., Printz, H., & Cohen, J. (1990). The effect of catecholamines on performance: From unit to system behavior. In D. Touretzky (Ed.), *Neural information processing systems 2.* San Mateo, CA: Morgan Kaufman, 100–108.

Stuss, D. & Benson, D. (1986). *The frontal lobes*. New York: Raven Press.

Zipser, D. (1991). Recurrent network model of the neural mechanism of short-term active memory. *Neural Computation, 3*, 179–193.
